# Retinal Processing Emulation in a Programmable 2-Layer Analog Array Processor CMOS Chip

**R. Carmona, F. Jiménez-Garrido, R. Domínguez-Castro,**
**S. Espejo, A. Rodríguez-Vázquez**
Instituto de Microelectrónica de Sevilla-CNM-CSIC
Avda. Reina Mercedes s/n 41012 Sevilla (SPAIN)
*rcarmona@imse.cnm.es*

## Abstract

A bio-inspired model for an analog programmable array processor (APAP), based on studies on the vertebrate retina, has permitted the realization of complex programmable spatio-temporal dynamics in VLSI. This model mimics the way in which images are processed in the visual pathway, rendering a feasible alternative for the implementation of early vision applications in standard technologies. A prototype chip has been designed and fabricated in a $0.5\mu$m standard CMOS process. Computing power per area and power consumption is amongst the highest reported for a single chip. Design challenges, trade-offs and some experimental results are presented in this paper.

## 1 Introduction

The conventional role of analog circuits in mixed-signal VLSI is providing the I/O interface to the digital core of the chip —which realizes all the signal processing. However, this approach may not be optimum for the processing of multi-dimensional sensory signals, such as those found in vision applications. When massive information flows have to be treated in parallel, it may be advantageous to realize some preprocessing in the analog domain, at the plane where signals are captured.

During the last years, different authors have focused on the realization of parallel preprocessing of multi-dimensional signals, using either purely digital techniques [1] or mixed-signal techniques, like in [2]. The data in Table 1 can help us to compare these two approaches. Here, the peak computing power (expressed as operations per second: XPS) per unit area and power is shown. This estimation is realized by considering the number of arithmetic analog operations that take place per unit time, in the analog case, or digital instructions per unit time, in the digital case. It can be seen that the computing power per area featured by chips based in Analog Programmable Array Processors (APAPs) is much higher than that exhibited by digital array processors. It can be argued that digital processors feature a larger accuracy, but accuracy requirements for vision applications are not rarely below 6

Table 1: Parallel processors comparison

| Reference | CMOS process | No. of cells | Cells/ mm$^2$ | XPS/ mm$^2$ | XPS/ mW |
|---|---|---|---|---|---|
| Liñan et. al. [2] | 0.5$\mu$m | 4096 | 81.0 | 7.93G | 0.33G |
| Gealow et. al. [1] | 0.6$\mu$m | 4096 | 66.7 | 4.00M | 1.00M |
| This chip | 0.5$\mu$m | 1024 | 29.2 | 6.01G | 1.56G |

bits. Also, taking full advantage of the full digital resolution requires highly accurate A/D converters, what creates additional area and power overhead.

The third row in Table 1 corresponds to the chip presented here. This chip outperforms the one in [2] in terms of functionality as it implements a reduced model of the biological retina [3]. It is capable of generating complex spatio-temporal dynamic processes, in a fully programmable way and with the possibility of storing intermediate processing results.

## 2 APAP chip architecture

### 2.1 Bio-inspired APAP model

The vertebrate retina has a layered structure [3], consisting, roughly, in a layer of photodetectors at the top, bipolar cells carrying signals across the retina, affected by the operation of horizontal and amacrine cells, and ganglion cells in the other end. There are, in this description, some interesting aspects that markedly resemble the characteristics of the Cellular Neural Networks (CNNs) [4]: 2D aggregations of continuous signals, local connectivity between elementary nonlinear processors, analog weighted interactions between them. Motivated by these coincidences, a model consisting of 2 layers of processors coupled by some inter-layer weights, and an additional layer incorporating analog arithmetics, has been developed [5]. Complex dynamics can be programmed via the intra- and inter-layer coupling strengths and the relation between the time constants of the layers. The evolution of each cell, $C(i,j)$, is described by two coupled differential equations, one for each CNN node:

$$\tau_n \frac{dx_{n,ij}}{dt} = -g[x_{n,ij}(t)] + \sum_{k=-r_1}^{r_1} \sum_{l=-r_1}^{r_1} a_{nn,kl} \cdot y_{n,(i+k)(j+l)} +$$
$$+ b_{nn,00} \cdot u_{nn,ij} + z_{n,ij} + a_{no} \cdot y_{no,ij} \qquad (1)$$

where $n$ and $o$ stand for the node in question and the other node respectively. The nonlinear losses term and the output function in each layer are those described for the full-signal range (FSR) model of the CNN [7], in which the state voltage is also limited and can be identified with the output voltage:

$$g(x_{n,ij}) = \lim_{m \to \infty} \begin{cases} m(x_{n,ij} - 1) + 1 & \text{if} \quad x_{n,ij} > 1 \\ x_{n,ij} & \text{if} \quad |x_{n,ij}| \leq 1 \\ -m(x_{n,ij} + 1) - 1 & \text{if} \quad x_{n,ij} < -1 \end{cases} \qquad (2)$$

and:

$$y_{n,ij} = f(x_{n,ij}) = \frac{1}{2}(|x_{n,ij} + 1| - |x_{n,ij} - 1|) \qquad (3)$$

The proposed chip consists in an APAP of $32 \times 32$ identical 2nd-order CNN cells (Fig. 3), surrounded by the circuits implementing the boundary conditions.

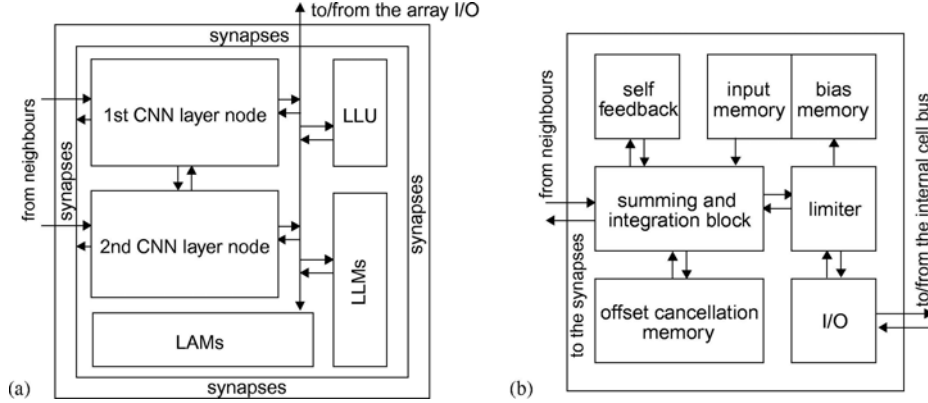

Figure 1: (a) Conceptual diagram of the basic cell and (b) internal structure of each CNN layer node

## 2.2  Basic processing cell architecture

Each elementary processor includes two coupled continuous-time CNN cores (Fig. 1(a)). The synaptic connections between processing elements of the same or different layer are represented by arrows in the diagram. The basic processor contains also a programmable local logic unit (LLU) and local analog and logic memories (LAMs and LLMs) to store intermediate results. The blocks in the cell communicate via an intra-cell data bus, multiplexed to the array interface. Control bits and switch configuration are passed to the cell from a global programming unit.

The internal structure of each CNN core is depicted in the diagram of Fig. 1(b). Each core receives contributions from the rest of the processing nodes in the neighbourhood which are summed and integrated in the state capacitor. The two layers differ in that the first layer has a scalable time constant, controlled by the appropriate binary code, while the second layer has a fixed time constant. The evolution of the state variable is also driven by self-feedback and by the feedforward action of the stored input and bias patterns. There is a voltage limiter for implementing the FSR CNN model. Forcing the state voltage to remain between these limits allows for using it as the output voltage. Then the state variable, which is now the output, is transmitted in voltage form to the synaptic blocks, in the periphery of the cell, where weighted contributions to the neighbours' are generated. There is also a current memory that will be employed for cancellation of the offset of the synaptic blocks. Initialization of the state, input and/or bias voltages is done through a mesh of multiplexing analog switches that connect to the cell's internal data bus.

# 3 Analog building blocks for the basic cell

## 3.1 Single-transistor synapse

The synapse is a four-quadrant analog multiplier. Their inputs will be the cell state, or input, and the weight voltages, while the output will be the cell's current contribution to a neighbouring cell. It can be realized by a single transistor biased in the ohmic region [6]. For a PMOS with gate voltage $V_X = V_{x_0} + V_x$, and the p-diffusion terminals at $V_W = V_{w_0} + V_w$ and $V_w$, the drain-to-source current is:

$$I_o \approx -\beta_p V_w V_x - \beta_p V_w \left( V_{x_0} + |\hat{V}_{T_p}| - V_{w_0} - \frac{V_w}{2} \right) \qquad (4)$$

which is a four-quadrant multiplier with an offset term that is time-invariant —at least during the evolution of the network— and not depending on the state. This offset is eliminated in a calibration step, with a current memory.

For the synapse to operate properly, the input node of the CNN core, (L) in Fig. 2, must be kept at a constant voltage. This is achieved by a current conveyor. Any difference between the voltage at node (L) and the reference $V_{w_0}$ is amplified and the negative feedback corrects the deviation. Notice that a voltage offset in the amplifier results in an error of the same order. An offset cancellation mechanism is provided (Fig. 2).

## 3.2 S³I current memory

As it has been referred, the offset term of the synapse current must be removed for its output current to represent the result of a four-quadrant multiplication. For this purpose all the synapses are reset to $V_X = V_{x_o}$. Then the resulting current, which is the sum of the offset currents of all the synapses concurrently connected to the same node, is memorized. This value will be substracted on-line from the input current when the CNN loop is closed, resulting in a one-step cancellation of the errors of all the synapses. The validity of this method relies in the accuracy of the current memory. For instance, in this chip, the sum of all the contributions will range, for the applications for which it has been designed, from $18\mu A$ to $46\mu A$. On the other side, the maximum signal to be handled is $1\mu A$. If a signal resolution of 8b is pretended, then 0.5LSB = 2nA. Thus, our current memory must be able to distinguish 2nA out of $46\mu A$. This represents an equivalent resolution of 14.5b. In order to achieve such accuracy level, a S³I current memory is used. It is composed by three stages (Fig. 2), each one consisting in a switch, a capacitor and a transistor. $I_B$ is the current to be memorized. After memorization the only error left corresponds to the last stage.

## 3.3 Time-constant scaling

The differential equation that governs the evolution of the network (1) can be written as a sum of current contributions injected to the state capacitor. Scaling up/down this sum of currents is equivalent to scaling the capacitor and, thus, speeding up/down the network dynamics. Therefore, scaling the input current with the help of a current mirror, for instance, will have the effect of scaling the time-constant. A circuit for continuously adjusting the current gain of a mirror can be designed based on a regulated-Cascode current mirror in the ohmic region. But the strong dependence of the ohmic-region biased transistors on the power rail voltage causes mismatches in $\tau$ between cells in the same layer. An alternative to this is

a digitally programmable current mirror. It trades resolution in $\tau$ for robustness, hence, the mismatch between the time constants of the different cells is now fairly attenuated.

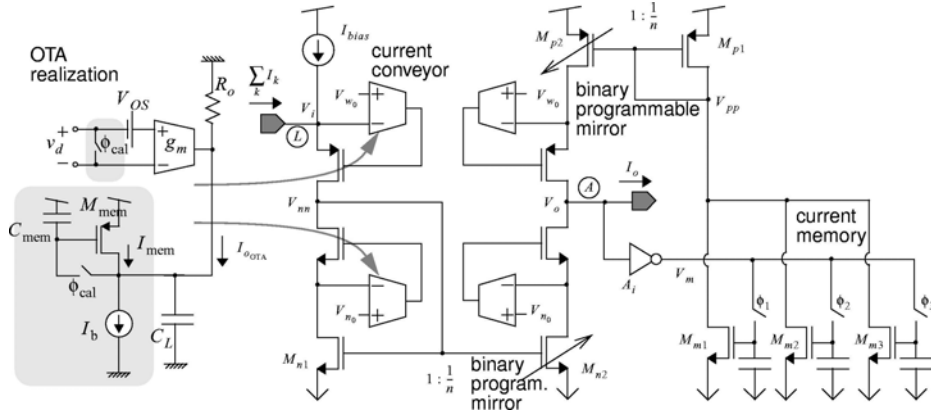

Figure 2: Input block with current scaling, S³I memory and offset-corrected OTA schematic

A new problem arises, though, because of current scaling. If the input current can be reshaped to a 16-times smaller waveform, then the current memory has to operate over a larger dynamic range. But, if designed to operate on large currents, the current memory will not work for the tiny currents of the scaled version of the input. If it is designed to run on small input currents, long transistors will be needed, and the operation will be unreliable for the larger currents. One way of avoiding this situation is to make the S³I memory to work on the original unscaled version of the input current. Therefore, the adjustable-time-constant CNN core will be a current conveyor, followed by the S³I current memory and then the binary weighted current mirror. The problem now is that the offsets introduced by the scaling block add up to the signal and the required accuracy levels can be lost. Our proposal is depicted in Fig. 2. It consists in placing the scaling block (programmable mirror) between the current conveyor and the current memory. In this way, any offset error will be cancelled in the auto-zeroing phase. In the picture, the voltage reference generated with the current conveyor, the regulated-Cascode current mirrors and the S³I memory can be easily identified. The inverter, $A_i$, driving the gates of the transistors of the current memory is required for stability.

## 4  Chip data and experimental results

A prototype chip has been designed and fabricated in a $0.5\mu$m single-poly triple-metal CMOS technology. Its dimensions are $9.27 \times 8.45\text{mm}^2$ (microphotograph in Fig. 3). The cell density achieved is $29.24\text{cells/mm}^2$, once the overhead circuitry is detracted from the total chip area —given that it does not scale linearly with the number of cells. The power consumption of the whole chip is around 300mW. Data I/O rates are nominally 10MS/s. Equivalent resolution for the analog images handled by the chip is 7.5 bit (measured). The time constant of the fastest layer (fixed time constant) is intended to be under 100ns. The peak computing power of this chip is, therefore, 470GXPS, what means $6.01\text{GXPS/mm}^2$, and $1.56\text{GXPS/mW}$.

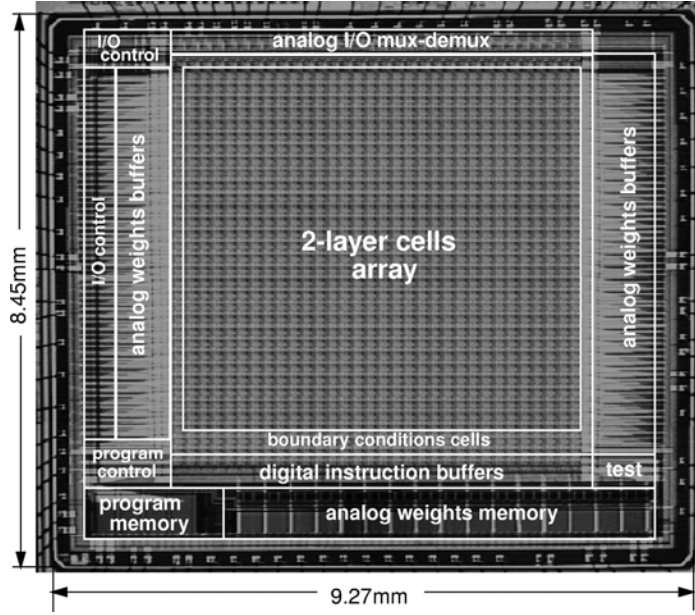

Figure 3: Prototype chip photograph

The programmable dynamics of the chip permit the observation of different phenomena of the type of propagation of active waves, pattern generation, etc. By tuning the coefficients that control the interactions between the cells in the array— i. e. the weights of the synaptic blocks, which are common to every elementary processor— different dynamics are manifested. Fig. 4 displays the evolution of the state variables of the two coupled layers when it is programmed to show different propagative behaviors. In picture (a), the chip is programmed to resemble the so-called wide-field erasure effect observed in the retina. Markers in the fastest layer (bottom row) trigger wavefronts in this layer and induce slower waves in the other layer (upper row). These induced spots are fedback, inhibiting the waves propagating in the fast layer, and generating a trailing edge for each wavefront. In picture (b), a solitary traveling wave is triggered from each corner of the fast layer. This kind of behavior is proper of waves in active media. Finally, in picture (c), edge detection is computed by extraction the low frequency components of the image, obtained by a diffusion in the slower layer, from theoriginal one. The remaining information is that of the higher frequency components of the image. These phenomena have been widely observed in measurements of the vertebrate retina [3]. They constitute the patterns of activity generated by the presence of visual stimuli.

Controlling the network dynamics and combining the results with the help of the built-in local logic and arithmetic operators, rather involved image processing tasks can be programmed like active-contour detection, object-tracking, etc.

## 5  Conclusions

From the figures obtained, we can state that the proposed approach supposes a promising alternative to conventional digital image processing for applications re-

lated with early-vision and low-level focal-plane image processing. Based on a simple but precise model of part of the real biological system, a feasible efficient implementation of an artificial vision device has been designed. The peak operation speed of the chip outperforms its digital counterparts due to the fully parallel nature of the processing. This especially so when comparing the computing power per silicon area unit and per watt.

## Acknowledgments

This work has been partially supported by ONR/NICOP Project N00014-00-1-0429, ESPRIT V Project IST-1999-19007, and by the Spanish CICYT Project TIC-1999-0826.

## References

[1] Gealow, J.C. & Sodini, C.G. (1999) A Pixel Parallel Image Processor Using Logic Pitch -Matched to Dynamic Memory. *IEEE Journal of Solid-State Circuits*, Vol. 34, No. 6, pp. 831-839.

[2] Liñan, G., Espejo, S., Domínguez-Castro, R., Roca, E. and Rodríguez-Vázquez, A. (1998) A 64 x 64 CNN with Analog and Digital I/O. *Proceedings of the IEEE Int. Conf. on Electronics, Circuits and Systems*, pp. 203-206, Lisbon, Portugal.

[3] Werblin, F. (1991) Synaptic Connections, Receptive Fields and Patterns of Activity in the Tiger Salamander Retina. *Investigative Ophthalmology and Visual Science*, Vol. 32, No. 3, pp. 459-483.

[4] Werblin, F., Roska, T. and Chua, L.O. (1995) The Analogic Cellular Neural Network as a Bionic Eye. *International Journal of Circuit Theory and Applications*, Vol. 23, No. 6, pp. 541-69.

[5] Rekeczky, Cs., Serrano-Gotarredona, T., Roska, T. and Rodríguez-Vázquez, A. (2000) A Stored Program 2nd Order/3- Layer Complex Cell CNN-UM. *Proc. of the Sixth IEEE International Workshop on Cellular Neural Networks and their Applications*, pp. 219-224, Catania, Italy.

[6] Domínguez-Castro, R., Rodríguez-Vázquez, A., Espejo, S. and Carmona, R. (1998) Four-Quadrant One-Transistor Synapse for High Density CNN Implementations. *Proc. of the Fifth IEEE International Workshop on Cellular Neural Networks and their Applications*, pp. 243-248, London, UK.

[7] Espejo, S., Carmona, R. Carmona, Domínguez-Castro, R. and Rodríguez-Vázquez, A. (1996) A VLSI Oriented Continuous- Time CNN Model. *International Journal of Circuits Theory and Applications*, Vol. 24, No. 3, pp. 341-356, John Wiley and Sons Ed.

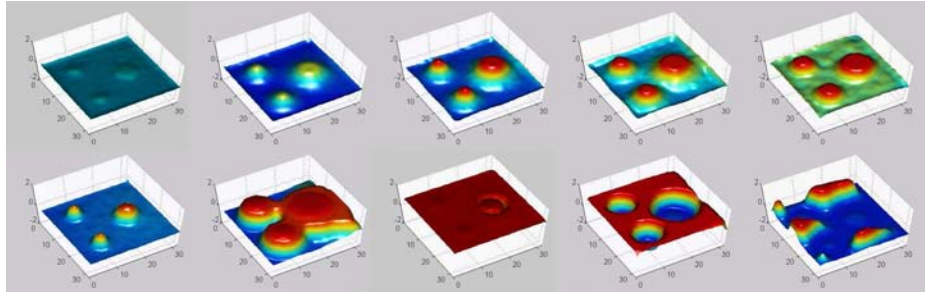

(a)

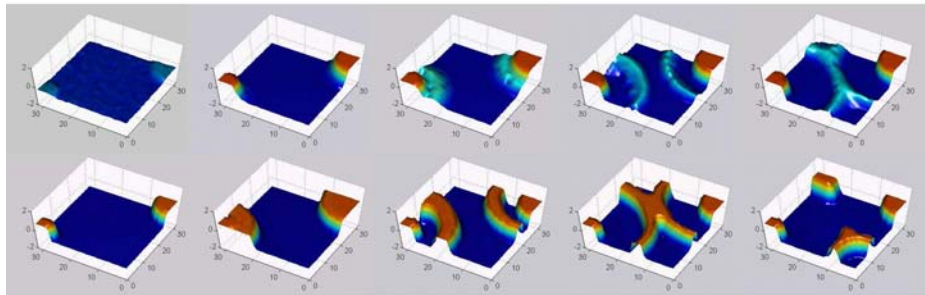

(b)

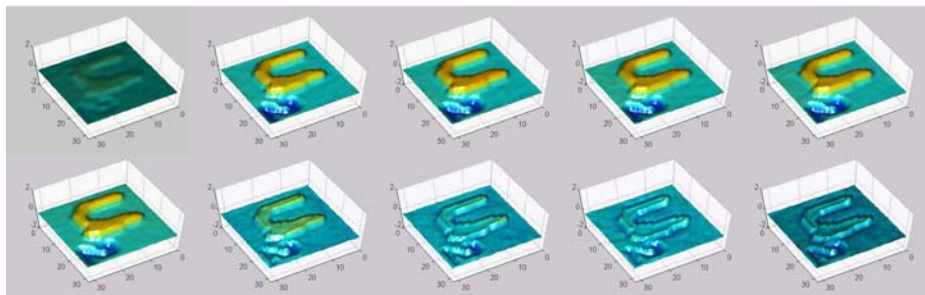

(c)

Figure 4: Examples of the different dynamics that can be programmed on the chip: (a) wide-field erasure effect, (b) traveling wave accross the layers, and (c) edge detection.